# ON PROPERTIES OF NETWORKS
# OF NEURON-LIKE ELEMENTS

Pierre Baldi[*] and Santosh S. Venkatesh[†]

15 December 1987

## Abstract

The complexity and computational capacity of multi-layered, feedforward neural networks is examined. Neural networks for special purpose (structured) functions are examined from the perspective of circuit complexity. Known results in complexity theory are applied to the special instance of neural network circuits, and in particular, classes of functions that can be implemented in shallow circuits characterised. Some conclusions are also drawn about learning complexity, and some open problems raised. The dual problem of determining the computational capacity of a class of multi-layered networks with dynamics regulated by an algebraic Hamiltonian is considered. Formal results are presented on the storage capacities of programmed higher-order structures, and a tradeoff between ease of programming and capacity is shown. A precise determination is made of the static fixed point structure of random higher-order constructs, and phase-transitions (0-1 laws) are shown.

## 1  INTRODUCTION

In this article we consider two aspects of computation with neural networks. Firstly we consider the problem of the complexity of the network required to compute classes of specified (structured) functions. We give a brief overview of basic known complexity theorems for readers familiar with neural network models but less familiar with circuit complexity theories. We argue that there is considerable computational and physiological justification for the thesis that shallow circuits (i.e., networks with relatively few layers) are computationally more efficient. We hence concentrate on structured (as opposed to random) problems that can be computed in shallow (constant depth) circuits with a relatively few number (polynomial) of elements, and demonstrate classes of structured problems that are amenable to such low cost solutions. We discuss an allied problem—the complexity of learning—and close with some open problems and a discussion of the observed limitations of the theoretical approach.

We next turn to a rigourous classification of how *much* a network of *given* structure can do; i.e., the computational capacity of a given construct. (This is, in

[*]Department of Mathematics, University of California (San Diego), La Jolla, CA 92093

[†]Moore School of Electrical Engineering, University of Pennsylvania, Philadelphia, PA 19104

a sense, the mirror image of the problem considered above, where we were seeking to design a minimal structure to perform a given task.) In this article we restrict ourselves to the analysis of higher-order neural structures obtained from polynomial threshold rules. We demonstrate that these higher-order networks are a special class of layered neural network, and present formal results on storage capacities for these constructs. Specifically, for the case of programmed interactions we demonstrate that the storage capacity is of the order of $n^d$ where $d$ is the interaction order. For the case of random interactions, a type of phase transition is observed in the distribution of fixed points as a function of attraction depth.

## 2  COMPLEXITY

There exist two broad classes of constraints on computations.

1. *Physical constraints*: These are related to the hardware in which the computation is embedded, and include among others time constants, energy limitations, volumes and geometrical relations in 3D space, and bandwidth capacities.

2. *Logical constraints*: These can be further subdivided into

    - Computability constraints—for instance, there exist unsolvable problems, i.e., functions such as the halting problem which are not computable in an absolute sense.

    - Complexity constraints—usually giving upper and/or lower bounds on the amount of resources such as the time, or the number of gates required to compute a given function. As an instance, the assertion "There exists an exponential time algorithm for the Traveling Salesman Problem," provides a computational upper bound.

If we view brains as computational devices, it is not unreasonable to think that in the course of the evolutionary process, nature may have been faced several times by problems related to physical and perhaps to a minor degree logical constraints on computations. If this is the case, then complexity theory in a broad sense could contribute in the future to our understanding of parallel computations and architectural issues both in natural and synthetic neural systems.

A simple theory of parallel processing at the macro level (where the elements are processors) can be developed based on the ratio of the time spent on communications between processors [7] for different classes of problems and different processor architecture and interconnections. However, this approach does not seem to work for parallel processing at the level of circuits, especially if calculations and communications are intricately entangled.

Recent neural or connectionist models are based on a common structure, that of highly interconnected networks of linear (or polynomial) threshold (or with sigmoid input-output function) units with adjustable interconnection weights. We shall therefore review the complexity theory of such circuits. In doing so, it will be sometimes helpful to contrast it with the similar theory based on Boolean (AND, OR, NOT) gates. The presentation will be rather informal and technical complements can easily be found in the references.

Consider a circuit as being on a cyclic oriented graph connecting $n$ Boolean inputs to one Boolean output. The nodes of the graph correspond to the gates (the $n$ input units, the "hidden" units, and the output unit) of the circuit. The *size* of the circuit is the total number of gates and the *depth* is the length of the longest path connecting one input to the output. For a layered, feed-forward circuit, the *width* is the average number of computational units in the hidden (or interior) layers of elements. The first obvious thing when comparing Boolean and threshold logic is that they are equivalent in the sense that any Boolean function can be implemented using either logic. In fact, any such function can be computed in a circuit of depth two and exponential size. Simple counting arguments show that the fraction of functions requiring a circuit of exponential size approaches one as $n \to \infty$ in both cases, i.e., a random function will in general require an exponential size circuit. (Paradoxically, it is very difficult to *construct* a family of functions for which we can prove that an exponential circuit is necessary.) Yet, threshold logic is more powerful than Boolean logic. A Boolean gate can compute only one function whereas a threshold gate can compute to the order of $2^{\alpha n^2}$ functions by varying the weights with $1/2 \le \alpha \le 1$ (see [19] for the lower bound; the upper bound is a classical hyperplane counting argument, see for instance [20,30]). It would hence appear plausible that there exist wide classes of problems which can be computed by threshold logic with circuits substantially smaller than those required by Boolean logic. An important result which separates threshold and Boolean logic from this point of view has been demonstrated by Yao [31] (see [10,24] for an elegant proof). The result is that in order to compute a function such as parity in a circuit of constant depth $k$, at least $\exp(cn^{1/2k})$ Boolean gates with unbounded fanin are required. As we shall demonstrate shortly, a circuit of depth two and linear size is sufficient for the computation of such functions using threshold logic.

It is not unusual to hear discussions about the tradeoffs between the depth and the width of a circuit. We believe that one of the main constristions of complexity analysis is to show that this tradeoff is in some sense minimal and that in fact there exists a very strong bias in favor of shallow (i.e., constant depth) circuits. There are multiple reasons for this. In general, for a fixed size, the number of different functions computable by a circuit of small depth exceeds the number of those computable by a deeper circuit. That is, if one had no a priori knowledge regarding the function to be computed and was given hidden units, then the optimal strategy would be to choose a circuit of depth two with the $m$ units in a single layer. In addition, if we view computations as propagating in a feedforward mode from the inputs to the output unit, then shallow circuits compute faster. And the deeper a circuit, the more difficult become the issues of time delays, synchronisation, and precision on the computations. Finally, it should be noticed that given overall responses of a few hundred milliseconds and given the known time scales for synaptic integration, biological circuitry must be shallow, at least within a "module" and this is corroborated by anatomical data. The relative slowness of neurons and their shallow circuit architecture are to be taken together with the "analog factor" and "entropy factor" [1] to understand the necessary high-connectivity requirements of neural systems.

From the previous analysis emerges an important class of circuits in threshold logic characterised by polynomial size and shallow depth. We have seen that, in general, a random function cannot be computed by such circuits. However, many interesting functions—the *structured problems*—are far from random, and it is then natural to ask what is the class of functions computable by such circuits? While a complete characterisation is probably difficult, there are several sub-classes of structural functions which are known to be computable in shallow poly-size circuits.

The *symmetric* functions, i.e., functions which are invariant under any permutation of the $n$ input variables, are an important class of structured problems that can be implemented in shallow polynomial size circuits. In fact, *any symmetric function can be computed by a threshold circuit of depth two and linear size*; ($n$ hidden units and one output unit are always sufficient). We demonstrate the validity of this assertion by the following instructive construction. We consider $n$ binary inputs, each taking on values -1 and 1 only, and threshold gates as units. Now array the $2^n$ possible inputs in $n + 1$ rows with the elements in each row being permuted versions of each other (i.e., $n$-tuples in a row all have the same number of +1's) and with the rows going monotonically from zero +1's to $n$ +1's. Any given symmetric Boolean function clearly assumes the same value for all elements (Boolean $n$-tuples) in a row, so that contiguous rows where the function assumes the value +1 form bands. (There are at most $n/2$ bands—the worst case occuring for the parity function.) The symmetric function can now be computed with $2B$ threshold gates in a single hidden layer with the topmost "neuron" being activated only if the number of +1's in the input exceeds the number of +1's in the lower edge of the lowest band, and proceeding systematically, the lowest "neuron" being activated only if the number of +1's in the input exceeds the number of +1's in the upper edge of the highest band. An input string will be within a band if and only if an odd number of hidden neurons are activated starting contiguously from the top of the hidden layer, and conversely. Hence, a single output unit can compute the given symmetric function.

It is easy to see that arithmetic operations on binary strings can be performed with polysize small depth circuits. Reif [23] has shown that for a fixed degree of precision, any analytic function such as polynomials, exponentials, and trigonometric functions can be approximated with small and shallow threshold circuits. Finally, in many situations one is interested in the value of a function only for a vanishingly small (i.e., polynomial) fraction of the total number of possible inputs $2^n$. These functions can be implemented by polysize shallow circuits and one can relate the size and depths of the circuit to the cardinal of the interesting inputs.

So far we only have been concerned with the complexity of threshold circuits. We now turn to the complexity of learning, i.e., the problem of finding the weights required to implement a given function. Consider the problem of repeating $m$ points in $\mathbb{R}^\ell$ coloured in two colours, using $k$ hyperplanes so that any region contains only monochromatic points. If $\ell$ and $k$ are fixed the problem can be solved in polynomial time. If either $\ell$ or $k$ goes to infinity, the problem becomes NP-complete [?]. As a result, it is not difficult to see that the general learning problem is NP-complete (see also [12] for a different proof and [21] for a proof of the fact it is already NP-complete in the case of one single threshold gate).

Some remarks on the limitations of the complexity approach are *a propos* at this juncture:

1. While a variety of structured Boolean functions can be implemented at relatively low cost with networks of linear threshold gates (McCulloch-Pitts neurons), the extension to different input-output functions and the continuous domain is not always straightforward.

2. Even restricting ourselves to networks of relatively simple Boolean devices such as the linear threshold gate, in many instances, only relatively weak bounds are available for computational cost and complexity.

3. Time is probably the single most important ingredient which is completely absent from these threshold units and their interconnections [17,14]; there are, in addition, non-biological aspects of connectionist models [8].

4. Finally, complexity results (where available) are often asymptotic in nature and may not be meaningful in the range corresponding to a particular application.

We shall end this section with a few open questions and speculations. One problem has to do with the time it takes to learn. Learning is often seen as a very slow process both in artificial models (cf. back propagation, for instance) and biological systems (cf. human acquisition of complex skills). However, if we follow the standards of complexity theory, in order to be effective over a wide variety of scales, a single learning algorithm should be polynomial time. We can therefore ask what is learnable by examples in polynomial time by polynomial size shallow threshold circuits? The status of back propagation type of algorithms with respect to this question is not very clear.

The existence of many tasks which are easily executed by biological organisms and for which no satisfactory computer program has been found so far leads to the question of the specificity of learning algorithms, i.e., whether there exists a complexity class of problems or functions for which a "program" can be found only by learning from examples as opposed to by traditional programming. There is some circumstantial evidence against such conjecture. As pointed out by Valiant [25], cryptography can be seen in some sense as the opposite of learning. The conjectures existence of one way function, i.e., functions which can be constructed in polynomial time but cannot be invested (from examples) in polynomial time suggests that learning algorithms may have strict limitations. In addition, for most of the artificial applications seen so far, the programs obtained through learning do not outperform the best already known software, though there may be many other reasons for that. However, even if such a complexity class does not exist, learning algorithm may still be very important because of their inexpensiveness and generality. The work of Valiant [26,13] on polynomial time learning of Boolean formulas in his "distribution free model" explores some additional limitations of what can be learned by examples without including any additional knowledge.

Learning may therefore turn out to be a powerful, inexpensive but limited family of algorithms that need to be incorporated as "sub-routines" of more global

programs, the structure of which may be harder to find. Should evolution be regarded as an "exponential" time learning process complemented by the "polynomial" time type of learning occurring in the lifetime of organisms?

## 3 CAPACITY

In the previous section the focus of our investigation was on the structure and cost of minimal networks that would compute specified Boolean functions. We now consider the dual question: What is the computational capacity of a threshold network of given structure? As with the issues on complexity, it turns out that for fairly general networks, the capacity results favour shallow (but perhaps broad) circuits [29]. In this discourse, however, we shall restrict ourselves to a specified class of higher-order networks, and to problems of associative memory. We will just quote the principal rigourous results here, and present the involved proofs elsewhere [4].

We consider systems of $n$ densely interacting threshold units each of which yields an instantaneous state -1 or +1. (This corresponds in the literature to a system of $n$ Ising spins, or alternatively, a system of $n$ neural states.) The state space is hence the set of vertices of the hypercube. We will in this discussion also restrict our attention throughout to *symmetric interaction systems* wherein the interconnections between threshold elements is bidirectional.

Let $\mathcal{I}_d$ be the family of all subsets of cardinality $d+1$ of the set $\{1, 2, \ldots, n\}$. Clearly $|\mathcal{I}_d| = \binom{n}{d+1}$. For any subset $I$ of $\{1, 2, \ldots, n\}$, and for every state $\mathbf{u} = \{u_1, u_2, \ldots, u_n\} \in \mathbb{B}^n \stackrel{\text{def}}{=} \{-1, 1\}^n$, set $u_I = \prod_{i \in I} u_i$.

**Definition 1** A *homogeneous algebraic threshold network* of degree $d$ is a network of $n$ threshold elements with interactions specified by a set of $\binom{n}{d+1}$ real coefficients $w_I$ indexed by $I$ in $\mathcal{I}_d$, and the evolution rule

$$u_i^+ = \text{sgn}\left(\sum_{I \in \mathcal{I}_d : i \in I} w_I u_{I \setminus \{i\}}\right) \tag{1}$$

These systems can be readily seen to be natural generalisations to higher-order of the familiar case $d = 1$ of *linear threshold networks*. The added degrees of freedom in the interaction coefficients can potentially result in enhanced flexibility and programming capability over the linear case as has been noted independently by several authors recently [2,3,4,5,22,27]. Note that each $d$-wise product $u_{I \setminus i}$ is just the parity of the corresponding $d$ inputs, and by our earlier discussion, this can be computed with $d$ hidden units in one layer followed by a single threshold unit. Thus the higher-order network can be realised by a network of depth three, where the first hidden layer has $d\binom{n}{d}$ units, the second hidden layer has $\binom{n}{d}$ units, and there are $n$ output units which feedback into the $n$ input units. Note that the weights from the input to the first hidden layer, and the first hidden layer to the second are fixed

(computing the various $d$-wise products), and the weights from the second hidden layer to the output are the coefficients $w_I$ which are free parameters.

These systems can be identified either with long range interactions for higher-order spin glasses at zero temperature, or higher-order neural networks. Starting from an arbitrary configuration or state, the system evolves asynchronously by a sequence of single "spin" flips involving spins which are misaligned with the instantaneous "molecular field." The dynamics of these symmetric higher-order systems are regulated analogous to the linear system by higher-order extensions of the classical quadratic Hamiltonian. We define the *homogeneous algebraic Hamiltonian* of degree $d$ by

$$H_d(\mathbf{u}) = - \sum_{I \in \mathcal{I}_d} w_I u_I \, . \tag{2}$$

The algebraic Hamiltonians are functionals akin in behaviour to the classical quadratic Hamiltonian as has been previously demonstrated [5].

**Proposition 1** The functional $H_d$ is non-increasing under the evolution rule 1.

In the terminology of spin glasses, the state trajectories of these higher-order networks can be seen to be following essentially a zero-temperature Monte Carlo (or Glauber) dynamics. Because of the monotonicity of the algebraic Hamiltonians given by equation 2 under the asynchronous evolution rule 1, the system always reaches a stable state (fixed point) where the relation 1 is satisfied for each of the $n$ spins or neural states. The fixed points are hence the arbiters of system dynamics, and determine the computational capacity of the system.

System behaviour and applications are somewhat different depending on whether the interactions are random or programmed. The case of random interactions lends itself to natural extensions of spin glass formulations, while programmed interactions yield applications of higher-order extensions of neural network models. We consider the two cases in turn.

## 3.1 PROGRAMMED INTERACTIONS

Here we query whether given sets of binary $n$-vectors can be stored as fixed points by a suitable selection of interaction coefficients. If such sets of *prescribed* vectors can be stored as stable states for some suitable choice of interaction coefficients, then proposition 1 will ensure that the chosen vectors are at the bottom of "energy wells" in the state space with each vector exercising a region of attraction around it—all characterestics of a physical associative memory. In such a situation the dynamical evolution of the network can be interpreted in terms of computations: error-correction, nearest neighbour search and associative memory. Of importance here is the maximum number of states that can be stored as fixed points for an appropriate choice of algebraic threshold network. This represents the *maximal information storage capacity* of such higher-order neural networks.

Let $d$ represent the degree of the algebraic threshold network. Let $\mathbf{u}^{(1)}, \ldots, \mathbf{u}^{(m)}$ be the $m$-set of vectors which we require to store as fixed points in a suitable algebraic threshold network. We will henceforth refer to these prescribed vectors as

*memories.* We define the *storage capacity* of an algebraic threshold network of degree $d$ to be the maximal number $m$ of arbitrarily chosen memories which can be stored with high probability for appropriate choices of coefficients in the network.

**Theorem 1** The maximal (algorithm independent) storage capacity of a homogeneous algebraic threshold network of degree $d$ is less than or equal to $2\binom{n}{d}$.

*Generalised Sum of Outer-Products Rule:* The classical Hebbian rule for the linear case $d = 1$ (cf. [11] and quoted references) can be naturally extended to networks of higher-order. The coefficients $w_I$, $I \in \mathcal{I}_d$ are constructed as the sum of generalised Kronecker outer-products,

$$w_I = \sum_{\alpha=1}^{m} u_I^{(\alpha)} .$$

**Theorem 2** The storage capacity of the outer-product algorithm applied to a homogeneous algebraic threshold network of degree $d$ is less than or equal to $n^d/2(d+1)\log n$ (also cf. [15,27]).

*Generalised Spectral Rule:* For $d = 1$ the spectral rule amounts to iteratively projecting states orthogonally onto the linear space generated by $\mathbf{u}^{(1)}, \ldots, \mathbf{u}^{(m)}$, and then taking the closest point on the hypercube to this projection (cf. [27,28]). This approach can be extended to higher-orders as we now describe.

Let $\mathbf{W}$ denote the $n \times N_{(n,d)}$ matrix of coefficients $w_I$ arranged lexicographically; i.e.,

$$\mathbf{W} = \begin{bmatrix} w_{1,1,2,\ldots,d-1,d} & w_{1,2,3,\ldots,d,d+1} & \cdots & w_{1,n-d+1,n-d+2,\ldots,n-1,n} \\ w_{2,1,2,\ldots,d-1,d} & w_{2,2,3,\ldots,d,d+1} & \cdots & w_{2,n-d+1,n-d+2,\ldots,n-1,n} \\ \vdots & \vdots & \vdots & \vdots \\ w_{n,1,2,\ldots,d-1,d} & w_{n,2,3,\ldots,d,d+1} & \cdots & w_{n,n-d+1,n-d+2,\ldots,n-1,n} \end{bmatrix}$$

Note that the symmetry and the "zero-diagonal" nature of the interactions have been relaxed to increase capacity. Let $\mathbf{U}$ be the $n \times m$ matrix of memories. Form the extended $N_{(n,d)} \times m$ binary matrix $^1\mathbf{U} = [^1\mathbf{u}^{(1)} \cdots {}^1\mathbf{u}^{(m)}]$, where

$$^1\mathbf{u}^{(\alpha)} = \begin{bmatrix} u_{1,2,\ldots,d-1,d}^{(\alpha)} \\ u_{1,2,\ldots,d-1,d+1}^{(\alpha)} \\ \vdots \\ u_{n-d+1,n-d+2,\ldots,n-1,n}^{(\alpha)} \end{bmatrix}$$

Let $\Lambda = \mathbf{dg}[\lambda^{(1)} \cdots \lambda^{(m)}]$ be a $m \times m$ diagonal matrix with positive diagonal terms. A generalisation of the spectral algorithm for choosing coefficients yields

$$\mathbf{W} = \mathbf{U}\Lambda^1\mathbf{U}^\dagger$$

where $^1\mathbf{U}^\dagger$ is the pseudo-inverse of $^1\mathbf{U}$.

**Theorem 3** The storage capacity of the generalised spectral algorithm is at best $\binom{n}{d}$.

## 3.2  RANDOM INTERACTIONS

We consider homogeneous algebraic threshold networks whose weights $w_I$ are i.i.d., $\mathcal{N}(0,1)$ random variables. This is a natural generalisation to higher-order of Ising spin glasses with Gaussian interactions. We will show an asymptotic estimate for the number of fixed points of the structure. Asymptotic results for the usual case $d = 1$ of linear threshold networks with Gaussian interactions have been reported in the literature [6,9,16].

For $i = 1, \ldots, n$ set

$$S_n^i = u_i \sum_{I \in \mathcal{I}_d : i \in I} w_I u_{I \setminus i} .$$

For each $n$ the random variables $S_n^i$, $i = 1, \ldots, n$ are identically distributed, jointly Gaussian variables with zero mean, and variance $\sigma_n^2 = \binom{n-1}{d}$.

**Definition 2** For any given $\beta \geq 0$, a state $\mathbf{u} \in \mathbb{B}^n$ is $\beta$-strongly stable iff $S_n^i \geq \beta \sigma_n$, for each $i = 1, \ldots, n$.

The case $\beta = 0$ reverts to the usual case of fixed points. The parameter $\beta$ is essentially a measure of how deep the well of attraction surrounding the fixed point is. The following proposition asserts that a 0-1 law ("phase transition") governs the expected number of fixed points which have wells of attraction above a certain depth. Let $F_d(\beta)$ be the expected number of $\beta$-strongly stable states.

**Theorem 4** Corresponding to each fixed interaction order $d$ there exists a positive constant $\beta_d^*$ such that as $n \to \infty$,

$$F_d^\beta \sim \begin{cases} k_d(\beta)\, 2^{n c_d(\beta)} & \text{if } \beta < \beta_d^* \\ k_d(\beta_d^*) & \text{if } \beta = \beta_d^* \\ 0 & \text{if } \beta > \beta_d^* , \end{cases}$$

where $k_d(\beta) > 0$, and $0 \leq c_d(\beta) < 1$ are parameters depending solely on $\beta$ and the interaction order $d$.

## 4  CONCLUSION

In fine, it appears possible to design shallow, polynomial size threshold circuits to compute a wide class of structured problems. The thesis that shallow circuits compute more efficiently than deep circuits is borne out. For the particular case of

higher-order networks, all the garnered results appear to point in the same direction: *For neural networks of fixed degree d, the maximal number of programmable states is essentially of the order of $n^d$.* The total number of fixed points, however, appear to be exponential in number (at least for the random interaction case) though almost all of them have constant attraction depths.

# References

[1] Y. S. Abu-Mostafa, "Number of synapses per neuron," in *Analog VLSI and Neural Systems*, ed. C. Mead, Addison Wesley, 1987.

[2] P. Baldi, *II. Some Contributions to the Theory of Neural Networks*. Ph.D. Thesis, California Insitute of Technology, June 1986.

[3] P. Baldi and S. S. Venkatesh, "Number of stable points for spin glasses and neural networks of higher orders," *Phys. Rev. Lett.*, vol. 58, pp. 913–916, 1987.

[4] P. Baldi and S. S. Venkatesh, "Fixed points of algebraic threshold networks," in preparation.

[5] H. H. Chen, et al, "Higher order correlation model of associative memory," in *Neural Networks for Computing*. New York: AIP Conf. Proc., vol. 151, 1986.

[6] S. F. Edwards and F. Tanaka, "Analytical theory of the ground state properties of a spin glass: I. ising spin glass," *Jnl. Phys. F*, vol. 10, pp. 2769–2778, 1980.

[7] G. C. Fox and S. W. Otto, "Concurrent Computations and the Theory of Complex Systems," *Caltech Concurrent Computation Program*, March 1986.

[8] F. H. Grick and C. Asanuma, "Certain aspects of the anatomy and physiology of the cerebral cortex," in *Parallel Distributed Processing*, vol. 2, eds. D. E. Rumelhart and J. L. McCelland, pp. 333–371, MIT Press, 1986.

[9] D. J. Gross and M. Mezard, "The simplest spin glass," *Nucl. Phys.*, vol. B240, pp. 431–452, 1984.

[10] J. Hasted, "Almost optimal lower bounds for small depth circuits," *Proc. 18-th ACM STOC*, pp. 6–20, 1986.

[11] J. J. Hopfield, "Neural networks and physical sytems with emergent collective computational abilities," *Proc. Natl. Acad. Sci. USA*, vol. 79, pp. 2554–2558, 1982.

[12] J. S. Judd, "Complexity of connectionist learning with various node functions," *Dept. of Computer and Information Science Technical Report*, vol. 87-60, Univ. of Massachussetts, Amherst, 1987.

[13] M. Kearns, M. Li, L. Pitt, and L. Valiant, "On the learnability of Boolean formulae," *Proc. 19-th ACM STOC*, 1987.

[14] C. Koch, T. Poggio, and V. Torre, "Retinal ganglion cells: A functional interpretation of dendritic morphology," *Phil. Trans. R. Soc. London*, vol. B 288, pp. 227–264, 1982.

[15] R. J. McEliece, E. C. Posner, E. R. Rodemich, and S. S. Venkatesh, "The capacity of the Hopfield associative memory," *IEEE Trans. Inform. Theory*, vol. IT–33, pp. 461–482, 1987.

[16] R. J. McEliece and E. C. Posner, "The number of stable points of an infinite-range spin glass memory," *JPL Telecomm. and Data Acquisition Progress Report*, vol. 42–83, pp. 209–215, 1985.

[17] C. A. Mead (ed.), *Analog VLSI and Neural Systems*, Addison Wesley, 1987.

[18] N. Megiddo, "On the complexity of polyhedral separability," to appear in *Jnl. Discrete and Computational Geometry*, 1987.

[19] S. Muroga, "Lower bounds on the number of threshold functions," *IEEE Trans. Elec. Comp.*, vol. 15, pp. 805–806, 1966.

[20] S. Muroga, *Threshold Logic and its Applications*, Wiley Interscience, 1971.

[21] V. N. Peled and B. Simeone, "Polynomial-time algorithms for regular set-covering and threshold synthesis," *Discr. Appl. Math.*, vol. 12, pp. 57–69, 1985.

[22] D. Psaltis and C. H. Park, "Nonlinear discriminant functions and associative memories," in *Neural Networks for Computing*. New York: AIP Conf. Proc., vol. 151, 1986.

[23] J. Reif, "On threshold circuits and polynomial computation," preprint.

[24] R. Smolenski, "Algebraic methods in the theory of lower bounds for Boolean circuit complexity," *Proc. 19-th ACM STOC*, 1987.

[25] L. G. Valiant, "A theory of the learnable," *Comm. ACM*, vol. 27, pp. 1134–1142, 1984.

[26] L. G. Valiant, "Deductive learning," *Phil. Trans. R. Soc. London*, vol. A 312, pp. 441–446, 1984.

[27] S. S. Venkatesh, *Linear Maps with Point Rules: Applications to Pattern Classification and Associative Memory*. Ph.D. Thesis, California Institute of Technology, Aug. 1986.

[28] S. S. Venkatesh and D. Psaltis, "Linear and logarithmic capacities in associative neural networks," to appear *IEEE Trans. Inform. Theory*.

[29] S. S. Venkatesh, D. Psaltis, and J. Yu, private communication.

[30] R. O. Winder, "Bounds on threshold gate realisability," *IRE Trans. Elec. Comp.*, vol. EC–12, pp. 561–564, 1963.

[31] A. C. C. Yao, "Separating the poly-time hierarchy by oracles," *Proc. 26-th IEEE FOCS*, pp. 1–10, 1985.
